# A Theoretical Analysis of Robust Coding over Noisy Overcomplete Channels

Eizaburo Doi[1], Doru C. Balcan[2], & Michael S. Lewicki[1,2]
[1]Center for the Neural Basis of Cognition,
[2]Computer Science Department,
Carnegie Mellon University, Pittsburgh, PA 15213
{edoi,dbalcan,lewicki}@cnbc.cmu.edu

## Abstract

Biological sensory systems are faced with the problem of encoding a high-fidelity sensory signal with a population of noisy, low-fidelity neurons. This problem can be expressed in information theoretic terms as coding and transmitting a multi-dimensional, analog signal over a set of noisy channels. Previously, we have shown that robust, overcomplete codes can be learned by minimizing the reconstruction error with a constraint on the channel capacity. Here, we present a theoretical analysis that characterizes the optimal linear coder and decoder for one- and two-dimensional data. The analysis allows for an arbitrary number of coding units, thus including both under- and over-complete representations, and provides a number of important insights into optimal coding strategies. In particular, we show how the form of the code adapts to the number of coding units and to different data and noise conditions to achieve robustness. We also report numerical solutions for robust coding of high-dimensional image data and show that these codes are substantially more robust compared against other image codes such as ICA and wavelets.

## 1   Introduction

In neural systems, the representational capacity of a single neuron is estimated to be as low as 1 bit/spike [1, 2]. The characteristics of the optimal coding strategy under such conditions, however, remains an open question. Recent efficient coding models for sensory coding such as sparse coding and ICA have provided many insights into visual sensory coding (for a review, see [3]), but those models made the implicit assumption that the representational capacity of individual neurons was infinite. Intuitively, such a limit on representational precision should strongly influence the form of the optimal code. In particular, it should be possible to increase the number of limited capacity units in a population to form a more precise representation of the sensory signal. However, to the best of our knowledge, such a code has not been characterized analytically, even in the simplest case.

Here we present a theoretical analysis of this problem for one- and two-dimensional data for arbitrary numbers of units. For simplicity, we assume that the encoder and decoder are both linear, and that the goal is to minimize the mean squared error (MSE) of the reconstruction. In contrast to our previous report, which examined noisy overcomplete

representations [4], the cost function does not contain a sparsity prior. This simplification makes the cost depend up to second order statistics, making it analytically tractable while preserving the robustness to noise.

## 2 The model

To define our model, we assume that the data is $N$-dimensional, has zero mean and covariance matrix $\Sigma_{\mathbf{x}}$, and define two matrices $\mathbf{W} \in \mathbb{R}^{M \times N}$ and $\mathbf{A} \in \mathbb{R}^{N \times M}$. For each data point $\mathbf{x}$, its representation $\mathbf{r}$ in the model is the linear transform of $\mathbf{x}$ through matrix $\mathbf{W}$, perturbed by the additive noise (i.e., channel noise) $\mathbf{n} \sim \mathcal{N}(\mathbf{0}, \sigma_n^2 \mathbf{I}_M)$:

$$\mathbf{r} = \mathbf{W}\mathbf{x} + \mathbf{n} = \mathbf{u} + \mathbf{n}. \tag{1}$$

We refer to $\mathbf{W}$ as the *encoding matrix* and its row vectors as *encoding vectors*. The reconstruction of a data point from its representation is simply the linear transform of the latter, using matrix $\mathbf{A}$:

$$\hat{\mathbf{x}} = \mathbf{A}\mathbf{r} = \mathbf{A}\mathbf{W}\mathbf{x} + \mathbf{A}\mathbf{n}. \tag{2}$$

We refer to $\mathbf{A}$ as the *decoding matrix* and its column vectors as *decoding vectors*. The term $\mathbf{A}\mathbf{W}\mathbf{x}$ in eq. 2 determines how the reconstruction depends on the data, while $\mathbf{A}\mathbf{n}$ reflects the channel noise in the reconstruction. When there is no channel noise ($\mathbf{n} = \mathbf{0}$), $\mathbf{A}\mathbf{W} = \mathbf{I}$ is equivalent to perfect reconstruction. A graphical description of this system is shown in Fig. 1.

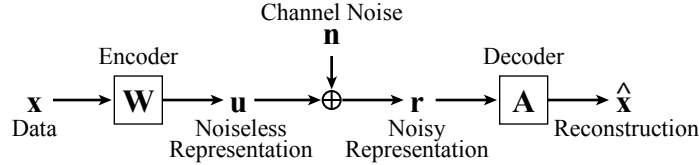

Figure 1: Diagram of the model.

The goal of the system is to form an accurate representation of the data that is robust to the presence of channel noise. We quantify the accuracy of the reconstruction by the mean squared error (MSE) over a set of data. The error of each sample is $\boldsymbol{\epsilon} = \mathbf{x} - \hat{\mathbf{x}} = (\mathbf{I}_N - \mathbf{A}\mathbf{W})\mathbf{x} - \mathbf{A}\mathbf{n}$, and the MSE is expressed in matrix form:

$$\mathcal{E}(\mathbf{A}, \mathbf{W}) = \text{tr}\{(\mathbf{I}_N - \mathbf{A}\mathbf{W})\Sigma_{\mathbf{x}}(\mathbf{I}_N - \mathbf{A}\mathbf{W})^T\} + \sigma_n^2 \text{tr}\{\mathbf{A}\mathbf{A}^T\}, \tag{3}$$

where we used $\mathcal{E} = \langle \boldsymbol{\epsilon}^T \boldsymbol{\epsilon} \rangle = \text{tr}(\langle \boldsymbol{\epsilon}\boldsymbol{\epsilon}^T \rangle)$. Note that, due to the MSE objective along with the zero-mean assumptions, the optimal solution depends solely on second-order statistics of the data and the noise.

Since the SNR is limited in the neural representation [1, 2], we assume that each coding unit has a limited variance $\langle u_i^2 \rangle = \sigma_u^2$ so that the SNR is limited to the same constant value $\gamma^2 = \sigma_u^2/\sigma_n^2$. As the channel capacity of information is defined by $C = \frac{1}{2}\ln(\gamma^2 + 1)$, this is equivalent to limiting the capacity of each unit to the same level. We will call this constraint as *channel capacity constraint*.

Now our problem is to minimize eq. 3 under the channel capacity constraint. To solve it, we will include this constraint in the parametrization of $\mathbf{W}$. Let $\Sigma_{\mathbf{x}} = \mathbf{E}\mathbf{D}\mathbf{E}^T$ be the eigenvalue decomposition of the data covariance matrix, and denote $\mathbf{S} = \mathbf{D}^{\frac{1}{2}} = \text{diag}(\sqrt{\lambda_1}, \cdots, \sqrt{\lambda_M})$, where $\lambda_i \equiv \mathbf{D}_{ii}$ are the $\Sigma_{\mathbf{x}}$'s eigenvalues. As we will see shortly, it is convenient to define $\mathbf{V} \equiv \mathbf{W}\mathbf{E}\mathbf{S}/\sigma_u$, then the condition $\langle u_i^2 \rangle = \sigma_u^2$ implies that

$$\mathbf{V}\mathbf{V}^T = \mathbf{C}_{\mathbf{u}} = \langle \mathbf{u}\mathbf{u}^T \rangle/\sigma_u^2, \tag{4}$$

where $\mathbf{C}_{\mathbf{u}}$ is the correlation matrix of the representation $\mathbf{u}$. Now the problem is formulated as a constrained optimization: finding the parameters that satisfy eq. 4 and minimize $\mathcal{E}$.

## 3 The optimal solutions and their characteristics

In this section we analyze the optimal solutions in some simple cases, namely for 1-dimensional (1-D) and 2-dimensional (2-D) data.

### 3.1 1-D data

In the 1-D case the MSE (eq. 3) is expressed as

$$\mathcal{E} = \sigma_x^2(1 - \mathbf{aw})^2 + \sigma_n^2\|\mathbf{a}\|_2^2, \tag{5}$$

where $\sigma_x^2 = \Sigma_\mathbf{x} \in \mathbb{R}$, $\mathbf{a} = \mathbf{A} \in \mathbb{R}^{1 \times M}$ and $\mathbf{w} = \mathbf{W} \in \mathbb{R}^{M \times 1}$. By solving the necessary condition for the minimum, $\partial\mathcal{E}/\partial\mathbf{a} = \mathbf{0}$, with the channel capacity constraint (eq. 4), the entries of the optimal solutions are

$$w_i = \pm\frac{\sigma_u}{\sigma_x}, \quad a_i = \frac{1}{w_i} \cdot \frac{\gamma^2}{M \cdot \gamma^2 + 1}, \tag{6}$$

and the smallest value of the MSE is

$$\mathcal{E} = \frac{\sigma_x^2}{M \cdot \gamma^2 + 1}. \tag{7}$$

This minimum depends on the SNR ($\gamma^2$) and on the number of units ($M$), and it is monotonically decreasing with respect to both. Furthermore, we can compensate for a decrease in SNR by an increase of the number of units. Note that $a_i$ are responsible for this adaptive behavior as $w_i$ do not vary with either $\gamma^2$ or $M$, in the 1-D case. The second term in eq. 5 leads the optimal $\mathbf{a}$ into having as small norm as possible, while the first term prevents it from being arbitrarily small. The optimum is given by the best trade-off between them.

### 3.2 2-D data

In the 2-D case, the channel capacity constraint (eq. 4) restricts $\mathbf{V}$ such that the row vectors of $\mathbf{V}$ should be on the unit circle. Therefore $\mathbf{V}$ can be parameterized as

$$\mathbf{V} = \begin{pmatrix} \cos\theta_1 & \sin\theta_1 \\ \vdots & \vdots \\ \cos\theta_M & \sin\theta_M \end{pmatrix}, \tag{8}$$

where $\theta_i \in [0, 2\pi)$ is the angle between $i$-th row of $\mathbf{V}$ and the principal eigenvector of the data $\mathbf{e}_1$ ($\mathbf{E} = [\mathbf{e}_1, \mathbf{e}_2]$, $\lambda_1 \geq \lambda_2 > 0$). The necessary condition for the minimum $\partial\mathcal{E}/\partial\mathbf{A} = \mathbf{O}$ implies

$$\mathbf{A} = \sigma_u\mathbf{E}\mathbf{S}\mathbf{V}^T(\sigma_u^2\mathbf{V}\mathbf{V}^T + \sigma_n^2\mathbf{I}_M)^{-1}. \tag{9}$$

Using eqs. 8 and 9, the MSE can be expressed as

$$\mathcal{E} = \frac{(\lambda_1 + \lambda_2)\left(\frac{2}{M}\gamma^2 + 1\right) - \frac{\gamma^2}{2}(\lambda_1 - \lambda_2)\operatorname{Re}(Z)}{\left(\frac{M}{2}\gamma^2 + 1\right)^2 - \frac{1}{4}\gamma^4|Z|^2}, \tag{10}$$

where by definition

$$Z = \sum_{k=1}^M z_k = \sum_{k=1}^M[\cos(2\theta_k) + i\sin(2\theta_k)]. \tag{11}$$

Now the problem has been reduced to finding simply a complex number $Z$ that minimizes $\mathcal{E}$. Note that $Z$ defines $\theta_k$ in $\mathbf{V}$, which in turn defines $\mathbf{W}$ (by definition; see eq. 4) and $\mathbf{A}$ (eq. 9). In the following we analyze the problem in two complementary cases: when the data variance is isotropic (i.e., $\lambda_1 = \lambda_2$), and when it is anisotropic ($\lambda_1 > \lambda_2$). As we will see, the solutions are qualitatively different in these two cases.

### 3.2.1 Isotropic case

Isotropy of the data variance implies $\lambda_1 = \lambda_2 \equiv \sigma_x^2$, and (without loss of generality) $\mathbf{E} = \mathbf{I}$, which simplifies the MSE (eq. 10) as

$$\mathcal{E} = \frac{2\sigma_x^2 \left(1 + \frac{M}{2}\gamma^2\right)}{\left(\frac{2}{M}\gamma^2 + 1\right)^2 - \frac{1}{4}\gamma^4 |Z|^2}. \tag{12}$$

Therefore, $\mathcal{E}$ is minimized whenever $|Z|^2$ is minimized.

If $M = 1$, $|Z|^2 = |z_1|^2$ is always 1 by definition (eq. 11), yielding the optimal solutions

$$\mathbf{W} = \frac{\sigma_u}{\sigma_x}\mathbf{V}, \quad \mathbf{A} = \frac{\sigma_x}{\sigma_u} \cdot \frac{\gamma^2}{\gamma^2 + 1}\mathbf{V}^T, \tag{13}$$

where $\mathbf{V} = \mathbf{V}(\theta_1)$, $\forall\,\theta_1 \in [0, 2\pi)$. Eq. 13 means that the orientation of the encoding and decoding vectors is arbitrary, and that the length of those vectors is adjusted exactly as in the 1-D case (eq. 6 with $M=1$; Fig. 2). The minimum MSE is given by

$$\mathcal{E} = \frac{\sigma_x^2}{\gamma^2 + 1} + \sigma_x^2. \tag{14}$$

The first term is the same as in the 1-D case (eq. 7 with $M=1$), corresponding to the error component along the axis that the encoding/decoding vectors represent, while the second term is the whole data variance along the axis orthogonal to the encoding/decoding vectors, along which no reconstruction is made.

If $M \geq 2$, there exists a set of angles $\theta_k$ for which $|Z|^2$ is 0. This can be verified by representing $Z$ in the complex plane (Z-diagram in Fig. 2) and observing that there is always a configuration of connected, unit-length bars that starts from, and ends up at the origin, thus indicating that $Z = |Z|^2 = 0$. Accordingly, the optimal solution is

$$\mathbf{W} = \frac{\sigma_u}{\sigma_x}\mathbf{V}, \quad \mathbf{A} = \frac{\sigma_x}{\sigma_u} \cdot \frac{\gamma^2}{\frac{M}{2}\gamma^2 + 1}\mathbf{V}^T, \tag{15}$$

where the optimal $\mathbf{V} = \mathbf{V}(\theta_1, \cdots, \theta_M)$ is given by such $\theta_1, \ldots, \theta_M$ for which $Z = 0$. Specifically, if $M=2$, then $z_1$ and $z_2$ must be antiparallel but are not otherwise constrained, making the pair of decoding vectors (and that of encoding vectors) orthogonal, yet free to rotate. Note that both the encoding and the decoding vectors are parallel to the rows of $\mathbf{V}$ (eq. 15), and the angle of $z_k$ from the real axis is twice as large as that of $\mathbf{a}_k$ (or $\mathbf{w}_k$). Likewise, if $M=3$, the decoding vectors should be evenly distributed yet still free to rotate; if $M=4$, the four vectors should just be two pairs of orthogonal vectors (not necessarily evenly distributed); if $M \geq 5$, there is no obvious regularity. With $Z = 0$, the MSE is minimized as

$$\mathcal{E} = \frac{2\sigma_x^2}{\frac{M}{2}\gamma^2 + 1}. \tag{16}$$

The minimum MSE (eq. 16) depends on the SNR ($\gamma^2$) and overcompleteness ratio ($M/N$) exactly in the same manner as explained in the 1-D case (eq. 7), considering that in both cases the numerator is the data variance, $\mathrm{tr}(\boldsymbol{\Sigma_x})$. We present examples in Fig 2: given $M = 2$, the reconstruction gets worse by lowering the SNR from 10 to 1; however, the reconstruction can be improved by increasing the number of units for a fixed SNR ($\gamma^2 = 1$). Just as in the 1-D case, the norm of the decoding vectors gets smaller by increasing $M$ or decreasing $\gamma^2$, which is explicitly described by eq. 15.

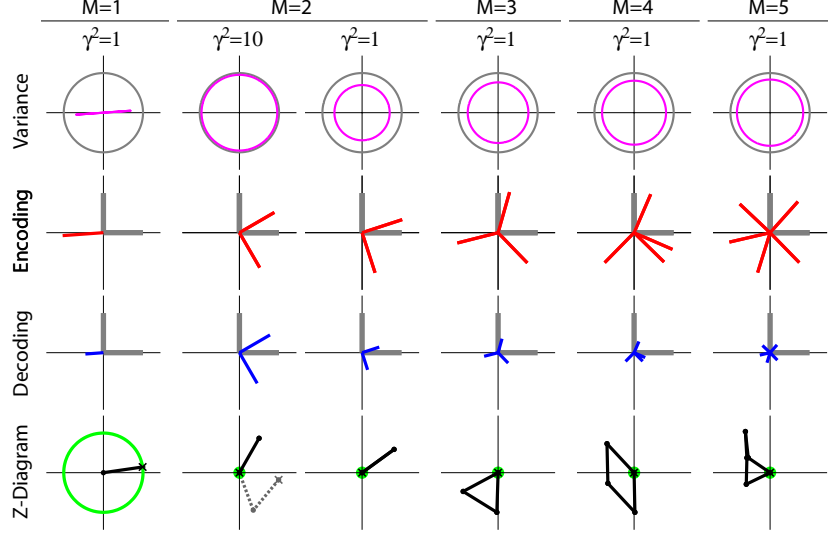

Figure 2: The optimal solutions for isotropic data. $M$ is the number of units and $\gamma^2$ is the SNR in the representation. "Variance" shows the variance ellipses for the data (gray) and the reconstruction (magenta). For perfect reconstruction, the two ellipses should overlap. "Encoding" and "Decoding" show encoding vectors (red) and decoding vectors (blue), respectively. The gray vectors show the principal axes of the data, $\mathbf{e}_1$ and $\mathbf{e}_2$. "Z-Diagram" represents $Z = \Sigma_k z_k$ (eq. 11) in the complex plane, where each unit length bar corresponds to a $z_k$, and the end point indicated by "$\times$" represents the coordinates of $Z$. The set of green dots in a plot corresponds to optimal values of $Z$; when this set reduces to a single dot, the optimal $Z$ is unique. In general there could be multiple configurations of bars for a single $Z$, implying multiple equivalent solutions of $\mathbf{A}$ and $\mathbf{W}$ for a given $Z$. For $M=2$ and $\gamma^2=10$, we drew with gray dotted bars an example of $Z$ that is not optimal (corresponding encoding and decoding vectors not shown).

### 3.2.2 Anisotropic case

In the anisotropic condition $\lambda_1 > \lambda_2$, the MSE (eq. 10) is minimized when $Z = \text{Re}(Z) \geq 0$ for a fixed value of $|Z|^2$. Therefore, the problem is reduced to seeking a real value $Z = y \in [0, M]$ that minimizes

$$\mathcal{E} = \frac{(\lambda_1 + \lambda_2)\left(\frac{M}{2}\gamma^2 + 1\right) - \frac{\gamma^2}{2}(\lambda_1 - \lambda_2)\,y}{\left(\frac{M}{2}\gamma^2 + 1\right)^2 - \frac{1}{4}\gamma^4 y^2}. \tag{17}$$

If $M=1$, then $y = \cos 2\theta_1$ from eq. 11, and therefore, $\mathcal{E}$ in eq. 17 is minimized iff $\theta_1 = 0$, yielding the optimal solutions

$$\mathbf{W} = \frac{\sigma_u}{\sqrt{\lambda_1}}\mathbf{e}_1^T, \quad \mathbf{A} = \frac{\sqrt{\lambda_1}}{\sigma_u}\cdot\frac{\gamma^2}{\gamma^2 + 1}\mathbf{e}_1. \tag{18}$$

In contrast to the isotropic case with $M=1$, the encoding and decoding vectors are specified along the principal axis ($\mathbf{e}_1$) as illustrated in Fig. 3. The minimum MSE is

$$\mathcal{E} = \frac{\lambda_1}{\gamma^2 + 1} + \lambda_2. \tag{19}$$

This is the same form as in the isotropic case (eq. 14) except that the first term is now related to the variance along the principal axis, $\lambda_1$, by which the encoding/decoding vectors can

most effectively be utilized for representing the data, while the second term is specified as the data variance along the minor axis, $\lambda_2$, by which the loss of reconstruction is mostly minimized. Note that it is a similar mechanism of dimensionality reduction as using PCA.

If $M \geq 2$, then we can derive the optimal $y$ from the necessary condition for the minimum, $d\mathcal{E}/dy = 0$, which yields

$$\left[\frac{\sqrt{\lambda_1} - \sqrt{\lambda_2}}{\sqrt{\lambda_1} + \sqrt{\lambda_2}}\left(M + \frac{2}{\gamma^2}\right) - y\right]\left[\frac{\sqrt{\lambda_1} + \sqrt{\lambda_2}}{\sqrt{\lambda_1} - \sqrt{\lambda_2}}\left(M + \frac{2}{\gamma^2}\right) - y\right] = 0. \tag{20}$$

Let $\gamma_c^2$ denote the SNR critical point, where

$$\gamma_c^2 = (\sqrt{\lambda_1/\lambda_2} - 1)/M. \tag{21}$$

If $\gamma^2 \geq \gamma_c^2$, then eq. 20 has a root within its domain $[0, M]$,

$$y = \frac{\sqrt{\lambda_1} - \sqrt{\lambda_2}}{\sqrt{\lambda_1} + \sqrt{\lambda_2}}\left(\frac{2}{\gamma^2} + M\right), \tag{22}$$

with $y = M$ if $\gamma^2 = \gamma_c^2$. Accordingly the optimal solutions are given by

$$\mathbf{W} = \mathbf{V}\left(\begin{array}{cc} \sigma_u/\sqrt{\lambda_1} & 0 \\ 0 & \sigma_u/\sqrt{\lambda_2} \end{array}\right)\mathbf{E}^T, \quad \mathbf{A} = \frac{\sqrt{\lambda_1} + \sqrt{\lambda_2}}{2\sigma_u} \cdot \frac{\gamma^2}{\frac{M}{2}\gamma^2 + 1}\mathbf{E}\mathbf{V}^T, \tag{23}$$

where the optimal $\mathbf{V} = \mathbf{V}(\theta_1, \cdots, \theta_M)$ is given by the Z-diagram as illustrated in Fig. 3, which we will describe shortly. The minimum MSE is given by

$$\mathcal{E} = \frac{1}{\frac{M}{2}\gamma^2 + 1}\frac{(\sqrt{\lambda_1} + \sqrt{\lambda_2})^2}{2}. \tag{24}$$

Note that eqs. 23–24 are reduced to eqs. 15–16 if $\lambda_1 = \lambda_2$.

If the SNR is smaller than $\gamma_c^2$, then $d\mathcal{E}/dy = 0$ does not have a root within the domain. However, $d\mathcal{E}/dy$ is always negative, and hence, $\mathcal{E}$ decreases monotonically on $[0, M]$. The minimum is therefore obtained when $y = M$, yielding the optimal solutions

$$\mathbf{W} = \frac{\sigma_u}{\sqrt{\lambda_1}}\mathbf{1}_M\mathbf{e}_1^T, \quad \mathbf{A} = \frac{\sqrt{\lambda_1}}{\sigma_u} \cdot \frac{\gamma^2}{M\gamma^2 + 1}\mathbf{e}_1\mathbf{1}_M^T, \tag{25}$$

where $\mathbf{1}_M = (1, \cdots, 1)^T \in \mathbb{R}^M$, and the minimum is given by

$$\mathcal{E} = \frac{\lambda_1}{M\gamma^2 + 1} + \lambda_2. \tag{26}$$

Note that $\mathcal{E}$ takes the same form as in $M = 1$ (eq. 19) except that we can now decrease the error by increasing the number of units. To summarize, if the representational resource is too limited either by $M$ or $\gamma^2$, the best strategy is to represent only the principal axis.

Now we describe the optimal solutions using the Z-diagram (Fig. 3). First, the optimal solutions differ depending on the SNR. If $\gamma^2 > \gamma_c^2$, the optimal $Z$ is a certain point between $0$ and $M$ on the real axis. Specifically, for $M = 2$ the optimal configuration of the unit-length connected bars is unique (up to flipping about x-axis), meaning that the encoding/decoding vectors are symmetric about the principal axis; for $M \geq 3$, there are infinitely many configurations of the bars starting from the origin and ending at the optimal $Z$, and nothing can be added about their regularity. If $\gamma^2 \leq \gamma_c^2$, the optimal $Z$ is $M$, and the optimal configuration is obtained only when all the bars align on the real axis. In this case, encoding/decoding vectors are all parallel to the principal axis ($\mathbf{e}_1$), as described by eq. 25. Such a degenerate representation is unique for the anisotropic case and is determined by $\gamma_c^2$ (eq. 21). We can

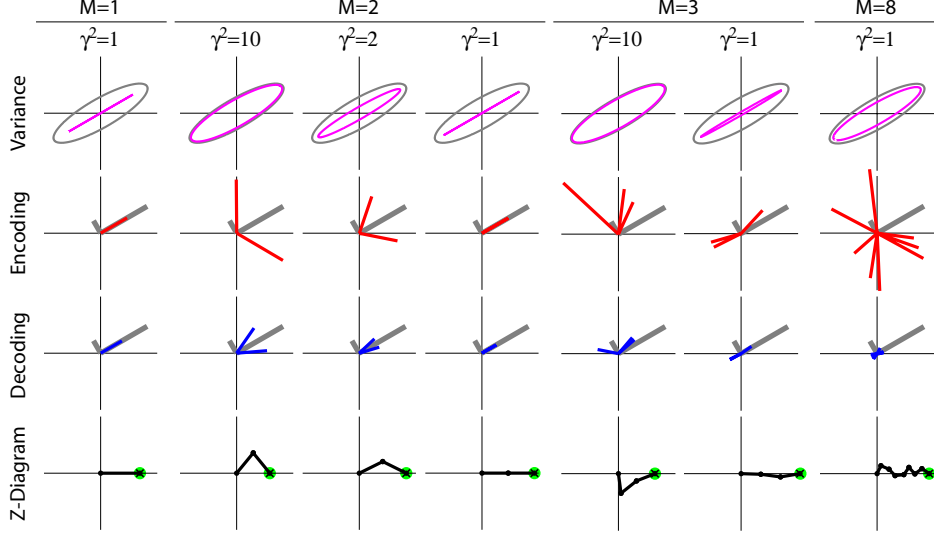

Figure 3: The optimal solutions for anisotropic data. Notations are as in Fig. 2. We set $\lambda_1 = 1.87$ and $\lambda_2 = 0.13$. $\gamma^2 > \gamma_c^2$ holds for all $M \geq 2$ but the one with $M=2$ and $\gamma^2 = 1$.

avoid the degeneration either by increasing the SNR (e.g., Fig. 3, $M=2$ with different $\gamma^2$) or by increasing the number of units ($\gamma^2 = 1$ with different $M$).

Also, the optimal solutions for the overcomplete representation are, in general, not obtained by simple replication (except in the degenerate case). For example, for $\gamma^2 = 1$ in Fig. 3, the optimal solution for $M=8$ is not identical to the replication of the optimal solution for $M=2$, and we can formally prove it by using eq. 22.

For $M=1$ and for the degenerate case, where only one axis in two dimensional space is represented, the optimal strategy is to preserve information along the principal axis at the cost of losing all information along the minor axis. Such a biased representation is also found for the non-degenerate case. We can see in Fig. 3 that the data along the principal axis is more accurately reconstructed than that along the minor axis; if there is no bias, the ellipse for the reconstruction should be similar to that of the data. More precisely, we can prove that the error ratio along $\mathbf{e}_1$ is smaller than that along $\mathbf{e}_2$ at the ratio of $\sqrt{\lambda_2} : \sqrt{\lambda_1}$ (note the switch of the subscripts), which describes the representation bias toward the main axis.

## 4 Application to image coding

In the case of high-dimensional data we can employ an algorithm similar to the one in [4], to numerically compute optimal solutions that minimizes the MSE subject to the channel capacity constraint. Fig. 4 presents the performance of our model when applied to image coding in the presence of channel noise. The data were $8 \times 8$ pixel blocks taken from a large image, and for comparison we considered representations with $M = 64$ ("1×") and respectively, $512$ ("8×") units. As for the channel capacity, each unit has 1.0 bit precision as in the neural representation [1]. The robust coding model shows a dramatic reduction in the reconstruction error, when compared to alternatives such as ICA and wavelet codes. This underscores the importance of taking into account the channel capacity constraint for better understanding the neural representation.

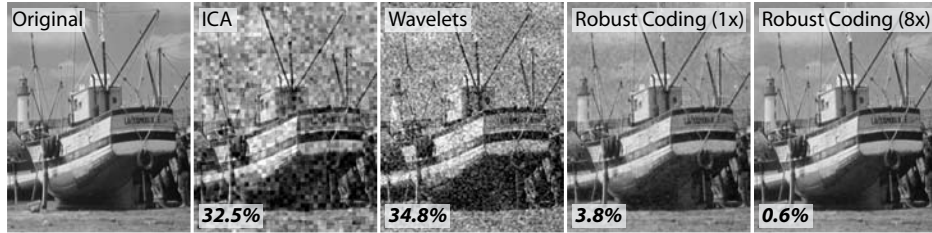

Figure 4: Reconstruction using one bit channel capacity representations. To ensure that all models had the same precision of 1.0 bit for each coefficient, we added Gaussian noise to the coefficients of the ICA and "Daubechies 9/7" wavelet codes as in the robust coding. For each representation, we displayed percentage error of the reconstruction. The results are consistent using other images, block size, or wavelet filters.

## 5    Discussion

In this study we measured the accuracy of the reconstruction by the MSE. An alternative measure could be, as in [5, 3], mutual information $I(\mathbf{x}, \hat{\mathbf{x}})$ between the data and the reconstruction. However, we can prove that this measure does not yield optimal solutions for the robust coding problem. Assuming the data is Gaussian and the representation is complete, we can prove that the mutual information is upper-bounded,

$$I(\mathbf{x}, \hat{\mathbf{x}}) \; = \; \frac{1}{2} \ln \det(\gamma^2 \mathbf{V}\mathbf{V}^T + \mathbf{I}_N) \; \leq \; \frac{N}{2} \ln(\gamma^2 + 1), \qquad (27)$$

with equality iff $\mathbf{V}\mathbf{V}^T = \mathbf{I}$, i.e., when the representation $\mathbf{u}$ is whitened (see eq. 4). This result holds even for anisotropic data, which is different from the optimal MSE code that can employ correlated, or even degenerate, representation. As ICA is one form of whitening, the results in Fig. 4 demonstrate the suboptimality of whitening in the MSE sense.

The optimal MSE code over noisy channels was examined previously in [6] for $N$-dimensional data. However, the capacity constraint was defined for a population and only examined the case of undercomplete codes. In the model studied here, motivated by the neural representation, the capacity constraint is imposed for individual units. Furthermore, the model allows for arbitrary number of units, which provides a way to arbitrarily improve the robustness of the code using a population code. The theoretical analysis for one- and two-dimensional cases quantifies the amount of error reduction as a function of the SNR and the number of units along with the data covariance matrix. Finally, our numerical results for higher-dimensional image data demonstrate a dramatic improvement in the robustness of the code over both conventional transforms such as wavelets and also representations optimized for statistical efficiency such as ICA.

## References

[1] A. Borst and F. E. Theunissen. Information theory and neural coding. *Nature Neuroscience*, 2:947–957, 1999.

[2] N. K. Dhingra and R. G. Smith. Spike generator limits efficiency of information transfer in a retinal ganglion cell. *Journal of Neuroscience*, 24:2914–2922, 2004.

[3] A. Hyvarinen, J. Karhunen, and E. Oja. *Independent Component Analysis*. Wiley, 2001.

[4] E. Doi and M. S. Lewicki. Sparse coding of natural images using an overcomplete set of limited capacity units. In *Advances in NIPS*, volume 17, pages 377–384. MIT Press, 2005.

[5] J. J. Atick and A. N. Redlich. What does the retina know about natural scenes? *Neural Computation*, 4:196–210, 1992.

[6] K. I. Diamantaras, K. Hornik, and M. G. Strintzis. Optimal linear compression under unreliable representation and robust PCA neural models. *IEEE Trans. Neur. Netw.*, 10(5):1186–1195, 1999.
